# Beyond Actions: Discriminative Models for Contextual Group Activities

**Tian Lan**
School of Computing Science
Simon Fraser University
tla58@sfu.ca

**Yang Wang**
Department of Computer Science
University of Illinois at Urbana-Champaign
yangwang@uiuc.edu

**Weilong Yang**
School of Computing Science
Simon Fraser University
wya16@sfu.ca

**Greg Mori**
School of Computing Science
Simon Fraser University
mori@cs.sfu.ca

## Abstract

We propose a discriminative model for recognizing group activities. Our model jointly captures the group activity, the individual person actions, and the interactions among them. Two new types of contextual information, *group-person interaction* and *person-person interaction*, are explored in a latent variable framework. Different from most of the previous latent structured models which assume a predefined structure for the hidden layer, e.g. a tree structure, we treat the structure of the hidden layer as a latent variable and implicitly infer it during learning and inference. Our experimental results demonstrate that by inferring this contextual information together with adaptive structures, the proposed model can significantly improve activity recognition performance.

## 1   Introduction

Look at the two persons in Fig. 1(a), can you tell they are doing two different actions? Once the entire contexts of these two images are revealed (Fig. 1(b)) and we observe the interaction of the person with other persons in the group, it is immediately clear that the first person is queuing, while the second person is talking. In this paper, we argue that actions of individual humans often cannot be inferred alone. We instead focus on developing methods for recognizing group activities by modeling the collective behaviors of individuals in the group.

Before we proceed, we first clarify some terminology used throughout the rest of the paper. We use *action* to denote a simple, atomic movement performed by a single person. We use *activity* to refer to a more complex scenario that involves a group of people. Consider the examples in Fig. 1(b), each frame describes a group activity: queuing and talking, while each person in a frame performs a lower level action: talking and facing right, talking and facing left, etc.

Our proposed approach is based on exploiting two types of contextual information in group activities. First, the activity of a group and the collective actions of all the individuals serve as context (we call it the *group-person interaction*) for each other, hence should be modeled jointly in a unified framework. As shown in Fig. 1, knowing the group activity (queuing or talking) helps disambiguate individual human actions which are otherwise hard to recognize. Similarly, knowing most of the persons in the scene are talking (whether facing right or left) allows us to infer the overall group activity (i.e. talking). Second, the action of an individual can also benefit from knowing the actions of other surrounding persons (which we call the *person-person interaction*). For example, consider Fig. 1(c). The fact that the first two persons are facing the same direction provides a strong cue that

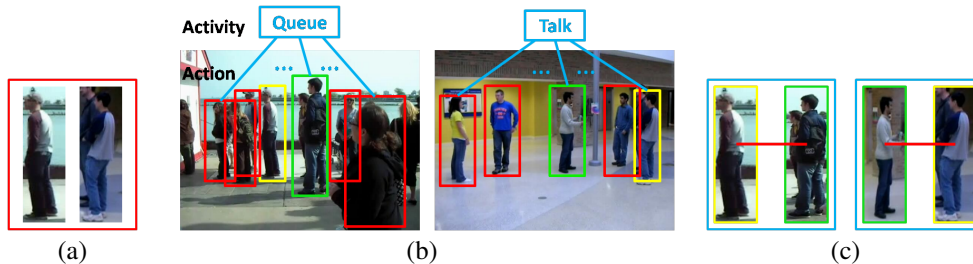

*Figure 1:* Role of context in group activities. It is often hard to distinguish actions from each individual person alone (a). However, if we look at the whole scene (b), we can easily recognize the activity of the group and the action of each individual. In this paper, we operationalize on this intuition and introduce a model for recognizing group activities by jointly consider the group activity, the action of each individual, and the interaction among certain pairs of individual actions (c).

both of them are queuing. Similarly, the fact that the last two persons are facing each other indicates they are more likely to be talking.

**Related work:** Using context to aid visual recognition has received much attention recently. Most of the work on context is in scene and object recognition. For example, work has been done on exploiting contextual information between scenes and objects [13], objects and objects [5, 16], objects and so-called "stuff" (amorphous spatial extent, e.g. trees, sky) [11], etc.

Most of the previous work in human action recognition focuses on recognizing actions performed by a single person in a video (e.g. [2, 17]). In this setting, there has been work on exploiting contexts provided by scenes [12] or objects [10] to help action recognition. In still image action recognition, object-action context [6, 9, 23, 24] is a popular type of context used for human-object interaction. The work in [3] is the closest to ours. In that work, person-person context is exploited by a new feature descriptor extracted from a person and its surrounding area.

Our model is directly inspired by some recent work on learning discriminative models that allow the use of latent variables [1, 6, 15, 19, 25], particularly when the latent variables have complex structures. These models have been successfully applied in many applications in computer vision, e.g. object detection [8, 18], action recognition [14, 19], human-object interaction [6], objects and attributes [21], human poses and actions [22], image region and tag correspondence [20], etc. So far only applications where the structures of latent variables are fixed have been considered, e.g. a tree-structure in [8, 19]. However in our applications, the structures of latent variables are not fixed and have to be inferred automatically.

**Our contributions:** In this paper, we develop a discriminative model for recognizing group activities. We highlight the main contributions of our model. (1) *Group activity:* most of the work in human activity understanding focuses on single-person action recognition. Instead, we present a model for group activities that dynamically decides on interactions among group members. (2) *Group-person and person-person interaction:* although contextual information has been exploited for visual recognition problems, ours introduces two new types of contextual information that have not been explored before. (3) *Adaptive structures:* the person-person interaction poses a challenging problem for both learning and inference. If we naively consider the interaction between every pair of persons, the model might try to enforce two persons to have take certain pairs of labels even though these two persons have nothing to do with each other. In addition, selecting a subset of connections allows one to remove "clutter" in the form of people performing irrelevant actions. Ideally, we would like to consider only those person-person interactions that are strong. To this end, we propose to use adaptive structures that automatically decide on whether the interaction of two persons should be considered. Our experimental results show that our adaptive structures significantly outperform other alternatives.

## 2 Contextual Representation of Group Activities

Our goal is to learn a model that jointly captures the group activity, the individual person actions, and the interactions among them. We introduce two new types of contextual information, *group-person*

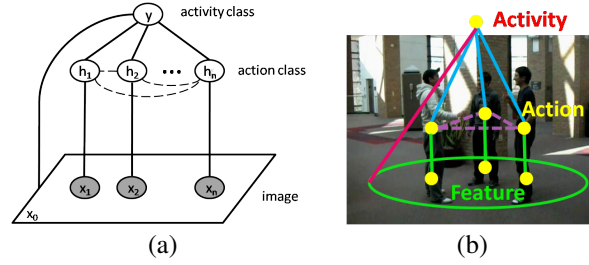

(a)                     (b)

*Figure 2:* Graphical illustration of the model in (a). The edges represented by dashed lines indicate the connections are latent. Different types of potentials are denoted by lines with different colors in the example shown in (b).

*interaction* and *person-person interaction*. Group-person interaction represents the co-occurrence between the activity of a group and the actions of all the individuals. Person-person interaction indicates that the action of an individual can benefit from knowing the actions of other people in the same scene. We present a graphical model representing all the information in a unified framework. One important difference between our model and previous work is that in addition to learning the parameters in the graphical model, we also automatically infer the graph structures (see Sec. 3).

We assume an image has been pre-processed (i.e. by running a person detector) so the persons in the image have been found. On the training data, each image is associated with a group activity label, and each person in the image is associated with an action label.

## 2.1   Model Formulation

A graphical representation of the model is shown in Fig. 2. We now describe how we model an image $I$. Let $I_1, I_2, \ldots, I_m$ be the set of persons found in the image $I$, we extract features $\mathbf{x}$ from the image $I$ in the form of $\mathbf{x} = (x_0, x_1, \ldots, x_m)$, where $x_0$ is the aggregation of feature descriptors of all the persons in the image (we call it *root feature vector*), and $x_i (i = 1, 2, \ldots, m)$ is the feature vector extracted from the person $I_i$. We denote the collective actions of all the persons in the image as $\mathbf{h} = (h_1, h_2, \ldots, h_m)$, where $h_i \in \mathcal{H}$ is the action label of the person $I_i$ and $\mathcal{H}$ is the set of all possible action labels. The image $I$ is associated with a group activity label $y \in \mathcal{Y}$, where $\mathcal{Y}$ is the set of all possible activity labels.

We assume there are connections between some pairs of action labels $(h_j, h_k)$. Intuitively speaking, this allows the model to capture important correlations between action labels. We use an undirected graph $\mathcal{G} = (\mathcal{V}, \mathcal{E})$ to represent $(h_1, h_2, \ldots, h_m)$, where a vertex $v_i \in \mathcal{V}$ corresponds to the action label $h_i$, and an edge $(v_j, v_k) \in \mathcal{E}$ corresponds to the interactions between $h_j$ and $h_k$.

We use $f_w(\mathbf{x}, \mathbf{h}, y; \mathcal{G})$ to denote the compatibility of the image feature $\mathbf{x}$, the collective action labels $\mathbf{h}$, the group activity label $y$, and the graph $\mathcal{G} = (\mathcal{V}, \mathcal{E})$. We assume $f_w(\mathbf{x}, \mathbf{h}, y; \mathcal{G})$ is parameterized by $w$ and is defined as follows:

$$f_w(\mathbf{x}, \mathbf{h}, y; \mathcal{G}) = w^\top \Psi(y, \mathbf{h}, \mathbf{x}; \mathcal{G}) \tag{1a}$$

$$= w_0^\top \phi_0(y, x_0) + \sum_{j \in \mathcal{V}} w_1^\top \phi_1(x_j, h_j) + \sum_{j \in \mathcal{V}} w_2^\top \phi_2(y, h_j) + \sum_{j,k \in \mathcal{E}} w_3^\top \phi_3(y, h_j, h_k) \tag{1b}$$

The model parameters $w$ are simply the combination of four parts, $w = \{w_1, w_2, w_3, w_4\}$. The details of the potential functions in Eq. 1 are described in the following:

**Image-Action Potential** $w_1^\top \phi_1(x_j, h_j)$: This potential function models the compatibility between the $j$-th person's action label $h_j$ and its image feature $x_j$. It is parameterized as:

$$w_1^\top \phi_1(x_j, h_j) = \sum_{b \in \mathcal{H}} w_{1b}^\top \mathbb{1}(h_j = b) \cdot x_j \tag{2}$$

where $x_j$ is the feature vector extracted from the $j$-th person and we use $\mathbb{1}(\cdot)$ to denote the indicator function. The parameter $w_1$ is simply the concatenation of $w_{1b}$ for all $b \in \mathcal{H}$.

**Action-Activity Potential** $w_2^\top \phi_2(y, h_j)$: This potential function models the compatibility between the group activity label $y$ and the $j$-th person's action label $h_j$. It is parameterized as:

$$w_2^\top \phi_2(y, h_j) = \sum_{a \in \mathcal{Y}} \sum_{b \in \mathcal{H}} w_{2ab} \cdot \mathbb{1}(y = a) \cdot \mathbb{1}(h_j = b) \tag{3}$$

**Action-Action Potential** $w_3^\top \phi_3(y, h_j, h_k)$: This potential function models the compatibility between a pair of individuals' action labels $(h_j, h_k)$ under the group activity label $y$, where $(j, k) \in \mathcal{E}$ corresponds to an edge in the graph. It is parameterized as:

$$w_3^\top \phi_3(y, h_j, h_k) = \sum_{a \in \mathcal{Y}} \sum_{b \in \mathcal{H}} \sum_{c \in \mathcal{H}} w_{3abc} \cdot \mathbb{1}(y = a) \cdot \mathbb{1}(h_j = b) \cdot \mathbb{1}(h_k = c) \tag{4}$$

**Image-Activity Potential** $w_0^\top \phi_0(y, x_0)$: This potential function is a root model which measures the compatibility between the activity label $y$ and the root feature vector $x_0$ of the whole image. It is parameterized as:

$$w_0^\top \phi_0(y, x_0) = \sum_{a \in \mathcal{Y}} w_{0a}^\top \, \mathbb{1}(y = a) \cdot x_0 \tag{5}$$

The parameter $w_{0a}$ can be interpreted as a root filter that measures the compatibility of the class label $a$ and the root feature vector $x_0$.

## 3 Learning and Inference

We now describe how to infer the label given the model parameters (Sec. 3.1), and how to learn the model parameters from a set of training data (Sec. 3.2). If the graph structure $\mathcal{G}$ is known and fixed, we can apply standard learning and inference techniques of latent SVMs. For our application, a good graph structure turns out to be crucial, since it determines which person interacts (i.e. provides action context) with another person. The interaction of individuals turns out to be important for group activity recognition, and fixing the interaction (i.e. graph structure) using heuristics does not work well. We will demonstrate this experimentally in Sec. 4. We instead develop our own inference and learning algorithms that automatically infer the best graph structure from a particular set.

### 3.1 Inference

Given the model parameters $w$, the inference problem is to find the best group activity label $y^*$ for a new image $\mathbf{x}$. Inspired by the latent SVM [8], we define the following function to score an image $\mathbf{x}$ and a group activity label $y$:

$$F_w(\mathbf{x}, y) = \max_{\mathcal{G}_y} \max_{\mathbf{h}_y} f_w(\mathbf{x}, \mathbf{h}_y, y; \mathcal{G}_y) = \max_{\mathcal{G}_y} \max_{\mathbf{h}_y} w^\top \Psi(\mathbf{x}, \mathbf{h}_y, y; \mathcal{G}_y) \tag{6}$$

We use the subscript $y$ in the notations $\mathbf{h}_y$ and $\mathcal{G}_y$ to emphasize that we are now fixing on a particular activity label $y$. The group activity label of the image $\mathbf{x}$ can be inferred as: $y^* = \arg\max_y F_w(\mathbf{x}, y)$. Since we can enumerate all the possible $y \in \mathcal{Y}$ and predict the activity label $y^*$ of $\mathbf{x}$, the main difficulty of solving the inference problem is the maximization over $\mathcal{G}_y$ and $\mathbf{h}_y$ according to Eq. 6. Note that in Eq. 6, we explicitly maximize over the graph $\mathcal{G}$. This is very different from previous work which typically assumes the graph structure is fixed.

The optimization problem in Eq. 6 is in general NP-hard since it involves a combinatorial search. We instead use an coordinate ascent style algorithm to approximately solve Eq. 6 by iterating the following two steps:

1. Holding the graph structure $\mathcal{G}_y$ fixed, optimize the action labels $\mathbf{h}_y$ for the $\langle \mathbf{x}, y \rangle$ pair:

$$\mathbf{h}_y = \arg\max_{\mathbf{h}'} w^\top \Psi(\mathbf{x}, \mathbf{h}', y; \mathcal{G}_y) \tag{7}$$

2. Holding $\mathbf{h}_y$ fixed, optimize graph structure $\mathcal{G}_y$ for the $\langle \mathbf{x}, y \rangle$ pair:

$$\mathcal{G}_y = \arg\max_{\mathcal{G}'} w^\top \Psi(\mathbf{x}, \mathbf{h}_y, y; \mathcal{G}') \tag{8}$$

The problem in Eq. 7 is a standard max-inference problem in an undirected graphical model. Here we use loopy belief propagation to approximately solve it. The problem in Eq. 8 is still an NP-hard problem since it involves enumerating all the possible graph structures. Even if we can enumerate all the graph structures, we might want to restrict ourselves to a subset of graph structures that will lead to efficient inference (e.g. when using loopy BP in Eq. 7). One obvious choice is to restrict $\mathcal{G}'$ to be a tree-structured graph, since loopy BP is exact and tractable for tree structured models. However, as we will demonstrate in Sec. 4, the tree-structured graph built from simple heuristic (e.g. minimum spanning tree) does not work that well. Another choice is to choose graph structures that are "sparse", since sparse graphs tend to have fewer cycles, and loopy BP tends to be efficient in graphs with fewer cycles. In this paper, we enforce the graph sparsity by setting a threshold $d$ on the maximum degree of any vertex in the graph. When $\mathbf{h}_y$ is fixed, we can formulate an integer linear program (ILP) to find the optimal graph structure (Eq. 8) with the additional constraint that the maximum vertex degree is at most $d$. Let $z_{jk} = 1$ indicate that the edge $(j, k)$ is included in the graph, and 0 otherwise. The ILP can be written as:

$$\max_{z} \sum_{j \in \mathcal{V}} \sum_{k \in \mathcal{V}} z_{jk} \psi_{jk}, \quad \text{s.t.} \quad \sum_{j \in \mathcal{V}} z_{jk} \leq d, \sum_{k \in \mathcal{V}} z_{jk} \leq d, \ z_{jk} = z_{kj}, \ z_{jk} \in \{0, 1\}, \ \forall j, k \quad (9)$$

where we use $\psi_{jk}$ to collectively represent the summation of all the pairwise potential functions in Eq. 1 for the pairs of vertices $(j, k)$. Of course, the optimization problem in Eq. 9 is still hard due to the integral constraint $z_{jk} \in \{0, 1\}$. But we can relax the value of $z_{jk}$ to a real value in the range of $[0, 1]$. The solution of the LP relaxation might have fractional numbers. To get integral solutions, we simply round them to the closest integers.

## 3.2   Learning

Given a set of N training examples $\langle \mathbf{x}^n, \mathbf{h}^n, y^n \rangle$ $(n = 1, 2, \ldots, N)$, we would like to train the model parameter $\mathbf{w}$ that tends to produce the correct group activity $y$ for a new test image $\mathbf{x}$. Note that the action labels $\mathbf{h}$ are observed on training data, but the graph structure $\mathcal{G}$ (or equivalently the variables $\mathbf{z}$) are unobserved and will be automatically inferred. A natural way of learning the model is to adopt the latent SVM formulation [8, 25] as follows:

$$\min_{w, \xi \geq 0, \mathcal{G}_y} \frac{1}{2} ||w||^2 + C \sum_{n=1}^{N} \xi_n \quad (10a)$$

$$\text{s.t.} \quad \max_{\mathcal{G}_{y^n}} f_w(\mathbf{x}^n, \mathbf{h}^n, y^n; \mathcal{G}_{y^n}) - \max_{\mathcal{G}_y} \max_{\mathbf{h}_y} f_w(\mathbf{x}^n, \mathbf{h}_y, y; \mathcal{G}_y) \geq \Delta(y, y^n) - \xi_n, \forall n, \forall y \quad (10b)$$

where $\Delta(y, y^n)$ is a loss function measuring the cost incurred by predicting $y$ when the ground-truth label is $y^n$. In standard multi-class classification problems, we typically use the 0-1 loss $\Delta_{0/1}$ defined as:

$$\Delta_{0/1}(y, y^n) = \begin{cases} 1 & \text{if } y \neq y^n \\ 0 & \text{otherwise} \end{cases} \quad (11)$$

The constrained optimization problem in Eq. 10 can be equivalently written as an unconstrained problem:

$$\min_{w, \xi} \quad \frac{1}{2} ||w||^2 + C \sum_{n=1}^{N} (\mathcal{L}^n - \mathcal{R}^n) \quad (12a)$$

where $\mathcal{L}^n = \max_{y} \max_{\mathbf{h}_y} \max_{\mathcal{G}_y} (\Delta(y, y^n) + f_w(\mathbf{x}^n, \mathbf{h}_y, y; \mathcal{G}_y))$, $\mathcal{R}^n = \max_{\mathcal{G}_{y^n}} f_w(\mathbf{x}^n, \mathbf{h}^n, y^n; \mathcal{G}_{y^n})$ (12b)

We use the non-convex bundle optimization in [7] to solve Eq. 12. In a nutshell, the algorithm iteratively builds an increasingly accurate piecewise quadratic approximation to the objective function. During each iteration, a new linear cutting plane is found via a subgradient of the objective function and added to the piecewise quadratic approximation. Now the key issue is to compute two subgradients $\partial_w \mathcal{L}^n$ and $\partial_w \mathcal{R}^n$ for a particular $w$, which we describe in detail below.

First we describe how to compute $\partial_w \mathcal{L}^n$. Let $(y^*, \mathbf{h}^*, \mathcal{G}^*)$ be the solution to the following optimization problem:

$$\max_{y} \max_{\mathbf{h}} \max_{\mathcal{G}} \Delta(y, y^n) + f_w(\mathbf{x}^n, \mathbf{h}, y; \mathcal{G}) \quad (13)$$

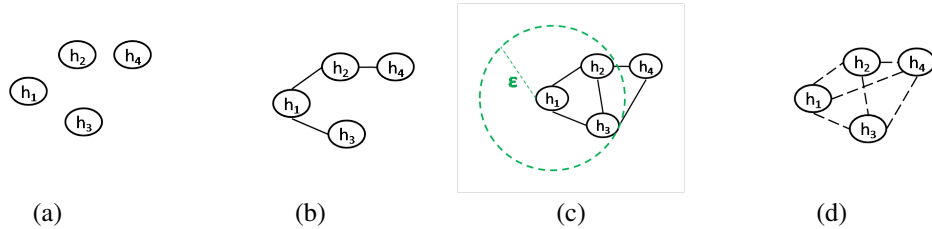

*Figure 3:* Different structures of person-person interaction. Each node here represents a person in a frame. Solid lines represent connections that can be obtained from heuristics. Dashed lines represent latent connections that will be inferred by our algorithm. (a) No connection between any pair of nodes; (b) Nodes are connected by a minimum spanning tree; (c) Any two nodes within a Euclidean distance $\varepsilon$ are connected (which we call *$\varepsilon$-neighborhood graph*); (d) Connections are obtained by adaptive structures. Note that (d) is the structure of person-person interaction of the proposed model.

Then it is easy to show that the subgradient $\partial_w \mathcal{L}^n$ can be calculated as $\partial_w \mathcal{L}^n = \Psi(\mathbf{x}^n, y^*, \mathbf{h}^*; \mathcal{G}^*)$. The inference problem in Eq. 13 is similar to the inference problem in Eq. 6, except for an additional term $\Delta(y, y^n)$. Since the number of possible choices of $y$ is small (e.g.$|\mathcal{Y}| = 5$) in our case), we can enumerate all possible $y \in \mathcal{Y}$ and solve the inference problem in Eq. 6 for each fixed $y$.

Now we describe how to compute $\partial_w \mathcal{R}^n$, let $\hat{\mathcal{G}}$ be the solution to the following optimization problem:

$$\max_{\mathcal{G}'} f_w(\mathbf{x}^n, \mathbf{h}^n, y^n; \mathcal{G}') \tag{14}$$

Then we can show that the subgradient $\partial_w \mathcal{R}^n$ can be calculated as $\partial_w \mathcal{R}^n = \Psi(\mathbf{x}^n, y^n, \mathbf{h}^n; \hat{\mathcal{G}})$. The problem in Eq. 14 can be approximately solved using the LP relaxation of Eq. 9. Using the two subgradients $\partial_w \mathcal{L}^n$ and $\partial_w \mathcal{R}^n$, we can optimize Eq. 10 using the algorithm in [7].

## 4 Experiments

We demonstrate our model on the collective activity dataset introduced in [3]. This dataset contains 44 video clips acquired using low resolution hand held cameras. In the original dataset, all the persons in every tenth frame of the videos are assigned one of the following five categories: *crossing, waiting, queuing, walking* and *talking*, and one of the following eight pose categories: *right, front-right, front, front-left, left, back-left, back* and *back-right*. Based on the original dataset, we define five activity categories including *crossing, waiting, queuing, walking* and *talking*. We define forty action labels by combining the pose and activity information, i.e. the action labels include *crossing and facing right, crossing and facing front-right,* etc. We assign each frame into one of the five activity categories, by taking the majority of actions of persons (ignoring their pose categories) in that frame. We select one fourth of the video clips from each activity category to form the test set, and the rest of the video clips are used for training.

Rather than directly using certain raw features (e.g. the HOG descriptor [4]) as the feature vector $x_i$ in our framework, we train a 40-class SVM classifier based on the HOG descriptor of each individual and their associated action labels. In the end, each feature vector $x_i$ is represented as a 40-dimensional vector, where the $k$-th entry of this vector is the score of classifying this instance to the $k$-th class returned by the SVM classifier. The root feature vector $x_0$ of an image is also represented as a 40-dimensional vector, which is obtained by taking an average over all the feature vectors $x_i$ ($i = 1, 2, ..., m$) in the same image.

**Results and Analysis:** In order to comprehensively evaluate the performance of the proposed model, we compare it with several baseline methods. The first baseline (which we call *global bag-of-words*) is a SVM model with linear kernel based on the global feature vector $x_0$ with a bag-of-words style representation. The other baselines are within our proposed framework, with various ways of setting the structures of the person-person interaction. The structures we have considered are illustrated in Fig. 3(a)-(c), including (a) no pairwise connection; (b) minimum spanning tree; (c) graph obtained by connecting any two vertices within a Euclidean distance $\varepsilon$ (*$\varepsilon$-neighborhood graph*) with $\varepsilon = 100, 200, 300$. Note that in our proposed model the person-person interactions are latent (shown in Fig. 3(d)) and learned automatically. The performance of different structures of person-person in-

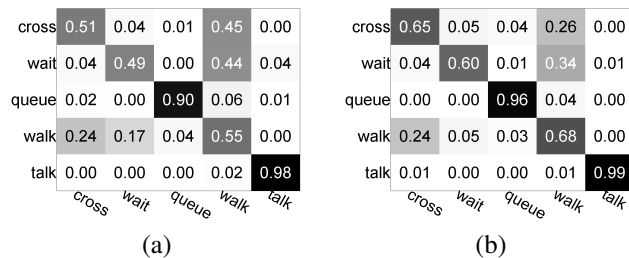

| | | | | | | | | | |
|---|---|---|---|---|---|---|---|---|---|
| cross | 0.51 | 0.04 | 0.01 | 0.45 | 0.00 | cross | 0.65 | 0.05 | 0.04 | 0.26 | 0.00 |

(a)                                                    (b)

*Figure 4:* Confusion matrices for activity classification: (a) global bag-of-words (b) our approach. Rows are ground-truths, and columns are predictions. Each row is normalized to sum to 1.

| Method | Overall | Mean per-class |
|:---:|:---:|:---:|
| global bag-of-words | 70.9 | 68.6 |
| no connection | 75.9 | 73.7 |
| minimum spanning tree | 73.6 | 70.0 |
| $\varepsilon$-neighborhood graph, $\varepsilon = 100$ | 74.3 | 72.9 |
| $\varepsilon$-neighborhood graph, $\varepsilon = 200$ | 70.4 | 66.2 |
| $\varepsilon$-neighborhood graph, $\varepsilon = 300$ | 62.2 | 62.5 |
| Our Approach | **79.1** | **77.5** |

*Table 1:* Comparison of activity classification accuracies of different methods. We report both the overall and mean per-class accuracies due to the class imbalance. The first result (global bag-of-words) is tested in the multi-class SVM framework, while the other results are in the framework of our proposed model but with different structures of person-person interaction. The structures are visualized in Fig. 3.

teraction are evaluated and compared. We summarize the comparison in Table 1. Since the test set is imbalanced, e.g. the number of crossing examples is more than twice that of the queuing or talking examples, we report both overall and mean per-class accuracies. As we can see, for both overall and mean per-class accuracies, our method achieves the best performance. The proposed model significantly outperforms *global bag-of-words*. The confusion matrices of our method and the baseline *global bag-of-words* are shown in Fig. 4. There are several important conclusions we can draw from these experimental results:

**Importance of group-person interaction**: The best result of the baselines comes from no connection between any pair of nodes, which clearly outperforms *global bag-of-words*. It demonstrates the effectiveness of modeling *group-person interaction*, i.e. connection between $y$ and $h$ in our model.

**Importance of adaptive structures of person-person interaction**: In Table 1, the pre-defined structures such as the minimum spanning tree and the *$\varepsilon$-neighborhood graph* do not perform as well as the one without person-person interaction. We believe this is because those pre-defined structures are all based on heuristics and are not properly integrated with the learning algorithm. As a result, they can create interactions that do not help (and sometimes even hurt) the performance. However, if we consider the graph structure as part of our model and directly infer it using our learning algorithm, we can make sure that the obtained structures are those useful for differentiating various activities. Evidence for this is provided by the big jump in terms of the performance by our approach.

We visualize the classification results and the learned structure of person-person interaction of our model in Fig. 6.

## 5   Conclusion

We have presented a discriminative model for group activity recognition which jointly captures the group activity, the individual person actions, and the interactions among them. We have exploited two new types of contextual information: *group-person interaction* and *person-person interaction*. We also introduce an adaptive structures algorithm that automatically infers the optimal structure of person-person interaction in a latent SVM framework. Our experimental results demonstrate that our proposed model outperforms other baseline methods.

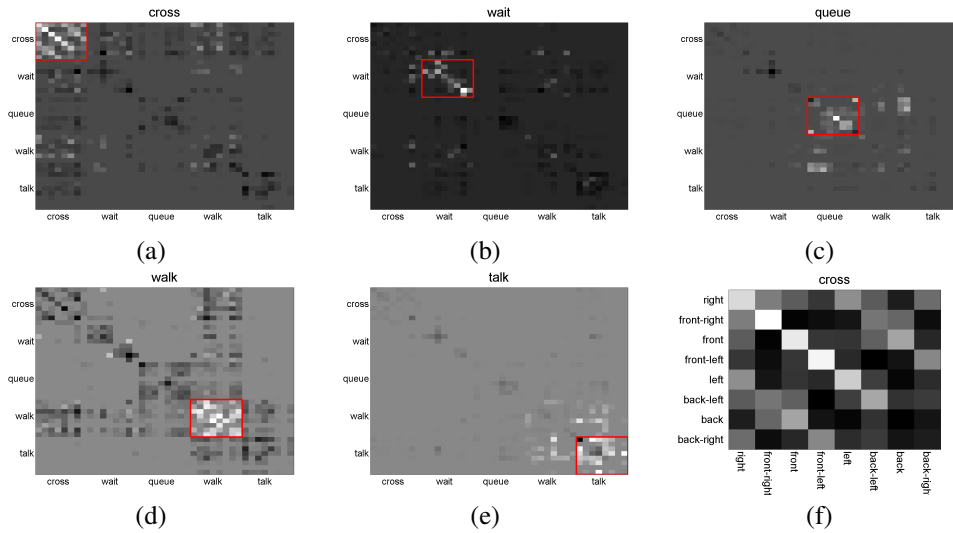

(a)          (b)          (c)

(d)          (e)          (f)

*Figure 5:* Visualization of the weights across pairs of action classes for each of the five activity classes. Light cells indicate large values of weights. Consider the example (a), under the activity label *crossing*, the model favors seeing actions of crossing with different poses together (indicated by the area bounded by the red box). We can also take a closer look at the weights within actions of crossing, as shown in (f). we can see that within the crossing category, the model favors seeing the same pose together, indicated by the light regions along the diagonal. It also favors some opposite poses, e.g. back-right with front-left. These make sense since people always cross street in either the same or the opposite directions.

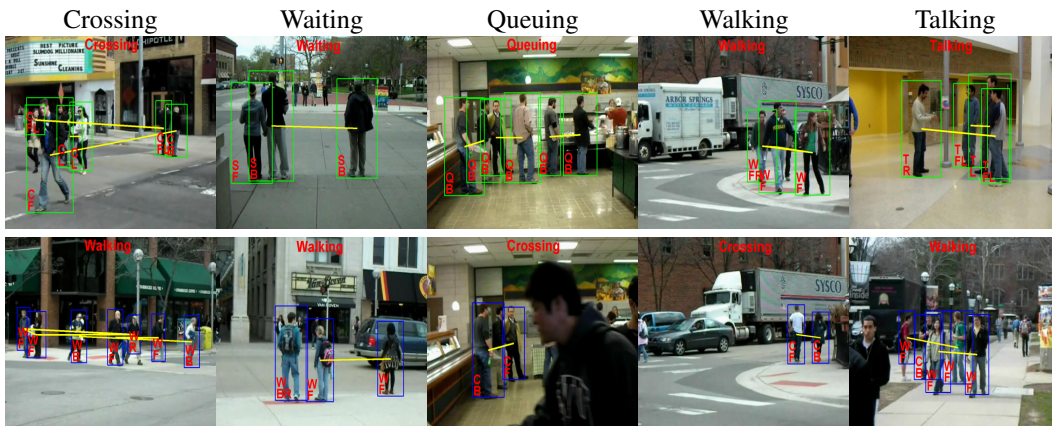

*Figure 6:* (Best viewed in color) Visualization of the classification results and the learned structure of person-person interaction. The top row shows correct classification examples and the bottom row shows incorrect examples. The labels C, S, Q, W, T indicate crossing, waiting, queuing, walking and talking respectively. The labels R, FR, F, FL, L, BL, B, BR indicate right, front-right, front, front-left, left, back-left, back and back-right respectively. The yellow lines represent the learned structure of person-person interaction, from which some important interactions for each activity can be obtained, e.g. a chain structure which connects persons facing the same direction is "important" for the *queuing* activity.

# References

[1] S. Andrews, I. Tsochantaridis, and T. Hofmann. Support vector machines for multiple-instance learning. In *Advances in Neural Information Processing Systems*, 2003.

[2] M. Blank, L. Gorelick, E. Shechtman, M. Irani, and R. Basri. Actions as space-time shapes. In *IEEE International Conference on Computer Vision*, 2005.

[3] W. Choi, K. Shahid, and S. Savarese. What are they doing? : Collective activity classification using spatio-temporal relationship among people. In *9th International Workshop on Visual Surveillance*, 2009.

[4] N. Dalal and B. Triggs. Histograms of oriented gradients for human detection. In *Proc. IEEE Comput. Soc. Conf. Comput. Vision and Pattern Recogn.*, 2005.

[5] C. Desai, D. Ramanan, and C. Fowlkes. Discriminative models for multi-class object layout. In *IEEE International Conference on Computer Vision*, 2009.

[6] C. Desai, D. Ramanan, and C. Fowlkes. Discriminative models for static human-object interactions. In *Workshop on Structured Models in Computer Vision*, 2010.

[7] T.-M.-T. Do and T. Artieres. Large margin training for hidden markov models with partially observed states. In *International Conference on Machine Learning*, 2009.

[8] P. Felzenszwalb, D. McAllester, and D. Ramanan. A discriminatively trained, multiscale, deformable part model. In *IEEE Computer Society Conference on Computer Vision and Pattern Recognition*, 2008.

[9] A. Gupta, A. Kembhavi, and L. S. Davis. Observing human-object interactions: Using spatial and functional compatibility for recognition. *IEEE Transactions on Pattern Analysis and Machine Intelligence*, 31(10):1775–1789, 2009.

[10] D. Han, L. Bo, and C. Sminchisescu. Selection and context for action recognition. In *IEEE International Conference on Computer Vision*, 2009.

[11] G. Heitz and D. Koller. Learning spatial context: Using stuff to find things. In *European Conference on Computer Vision*, 2008.

[12] M. Marszalek, I. Laptev, and C. Schmid. Actions in context. In *IEEE Computer Society Conference on Computer Vision and Pattern Recognition*, 2009.

[13] K. P. Murphy, A. Torralba, and W. T. Freeman. Using the forest to see the trees: A graphicsl model relating features, objects, and scenes. In *Advances in Neural Information Processing Systems*, volume 16. MIT Press, 2004.

[14] J. C. Niebles, C.-W. Chen, , and L. Fei-Fei. Modeling temporal structure of decomposable motion segments for activity classification. In *European Conference of Computer Vision*, 2010.

[15] A. Quattoni, S. Wang, L.-P. Morency, M. Collins, and T. Darrell. Hidden conditional random fields. *IEEE Transactions on Pattern Analysis and Machine Intelligence*, 29(10):1848–1852, June 2007.

[16] A. Rabinovich, A. Vedaldi, C. Galleguillos, E. Wiewiora, and S. Belongie. Objects in context. In *IEEE International Conference on Computer Vision*, 2007.

[17] C. Schuldt, I. Laptev, and B. Caputo. Recognizing human actions: A local svm approach. In *17th International Conference on Pattern Recognition*, 2004.

[18] A. Vedaldi and A. Zisserman. Structured output regression for detection with partial truncation. In *Advances in Neural Information Processing Systems*. MIT Press, 2009.

[19] Y. Wang and G. Mori. Max-margin hidden conditional random fields for human action recognition. In *Proc. IEEE Comput. Soc. Conf. Comput. Vision and Pattern Recogn.*, 2009.

[20] Y. Wang and G. Mori. A discriminative latent model of image region and object tag correspondence. In *Advances in Neural Information Processing Systems (NIPS)*, 2010.

[21] Y. Wang and G. Mori. A discriminative latent model of object classes and attributes. In *European Conference on Computer Vision*, 2010.

[22] W. Yang, Y. Wang, and G. Mori. Recognizing human actions from still images with latent poses. In *CVPR*, 2010.

[23] B. Yao and L. Fei-Fei. Grouplet: a structured image representation for recognizing human and object interactions. In *The Twenty-Third IEEE Conference on Computer Vision and Pattern Recognition*, San Francisco, CA, June 2010.

[24] B. Yao and L. Fei-Fei. Modeling mutual context of object and human pose in human-object interaction activities. In *The Twenty-Third IEEE Conference on Computer Vision and Pattern Recognition*, San Francisco, CA, June 2010.

[25] C.-N. Yu and T. Joachims. Learning structural SVMs with latent variables. In *International Conference on Machine Learning*, 2009.

